# On the Theory of Learning with Privileged Information

**Dmitry Pechyony**
NEC Laboratories
Princeton, NJ 08540, USA
pechyony@nec-labs.com

**Vladimir Vapnik**
NEC Laboratories
Princeton, NJ 08540, USA
vlad@nec-labs.com

## Abstract

In *Learning Using Privileged Information (LUPI)* paradigm, along with the standard training data in the decision space, a teacher supplies a learner with the privileged information in the correcting space. The goal of the learner is to find a classifier with a low generalization error in the decision space. We consider an empirical risk minimization algorithm, called Privileged ERM, that takes into account the privileged information in order to find a good function in the decision space. We outline the conditions on the correcting space that, if satisfied, allow Privileged ERM to have much faster learning rate in the decision space than the one of the regular empirical risk minimization.

## 1   Introduction

In the classical supervised machine learning paradigm the learner is given a labeled training set of examples and her goal is to find a decision function with the small generalization error on the unknown test examples. If the learning problem is easy (e.g. if learner's space of decision functions contains a one with zero generalization error) then, when the training size increases, the decision function found by the learner converges quickly to the optimal one. However if the learning problem is hard and the learner's space of decision functions is large then the convergence (or learning) rate is slow. The example of such hard learning problem is XOR when the space of decision functions is 2-dimensional hyperplanes.

The obvious question is "*Can we accelerate the learning rate if the learner is given an additional information about the learning problem?*". During the last years several new paradigms of learning with additional information were proposed that, under some conditions, provably accelerate the learning rate. For example, in semi-supervised learning such additional information is unlabeled training examples.

In this paper we consider a recently proposed *Learning Using Privileged Information (LUPI)* paradigm [8, 9, 10], that uses additional information of different kind. Let $X$ be a decision space. In LUPI paradigm, in addition to the standard training data, $(x, y) \in X \times Y$, a teacher supplies the learner with a *privileged information* $x^*$ in the correcting space $X^*$. The privileged information is only available for the training examples and is never available for the test examples. The LUPI paradigm requires, given a training set $\{(x_i, x_i^*, y_i)\}_{i=1}^n$, to find a decision function $h : X \to Y$ with the small generalization error for the unknown test examples $x \in X$.

The above question about accelerating the learning rate, reformulated in terms of the LUPI paradigm, is "*What kind of additional information should the teacher provide to the learner in order to accelerate her learning rate?*". Paraphrased, this question is essentially "*Who is a good teacher?*". In this paper we outline the conditions for the additional information provided by the teacher that allow for fast learning rate even in the hard problems.

LUPI paradigm emerges in a number of applications, for example time series prediction, protein classification and human computation. The experiments [9] in these domains demonstrated a clear advantage of LUPI paradigm over the supervised learning.

LUPI paradigm can be implemented by SVM+ algorithm [8], which in turn is based on the well-known SVM algorithm [2]. We now present the version of SVM+ for classification, the version for regression can be found in [9]. Let $h(x) = \mathrm{sign}(w \cdot x + b)$ be a decision function and $\phi(x_i^*) = w^* \cdot x_i^* + d$ be a *correcting function*. The optimization problem of SVM+ is

$$\min_{w,b,w^*,d} \quad \frac{1}{2}\|w\|_2^2 + \frac{\gamma}{2}\|w^*\|_2^2 + C\sum_{i=1}^{n}(w^* \cdot x_i^* + d) \tag{1}$$

$$\text{s.t. } \forall\, 1 \le i \le n, \quad y_i(w \cdot x_i + b) \ge 1 - (w^* \cdot x_i^* + d)$$
$$\forall\, 1 \le i \le n, \quad w^* \cdot x_i^* + d \ge 0.$$

The objective function of SVM+ contains two hyperparameters, $C > 0$ and $\gamma > 0$. The term $\gamma\|w^*\|_2^2/2$ in (1) is intended to restrict the capacity (or VC-dimension) of the function space containing $\phi$.

Let $\ell_X(h(x), y) = 1 - y(w \cdot x + b)$ be a hinge loss of the decision function $h = (w, b)$ on the example $(x, y)$ and $\ell_{X^*}(\phi(x^*)) = [w^* \cdot x^* + d]_+$ be a loss of the correcting function $\phi = (w^*, d)$ on the example $x^*$. The optimization problem (1) can be rewritten as

$$\min_{h=(w,b),\phi=(w^*,d)} \quad \frac{1}{2}\|w\|_2^2 + \frac{\gamma}{2}\|w^*\|_2^2 + C\sum_{i=1}^{n}\ell_{X^*}(\phi(x_i^*)) \tag{2}$$

$$\text{s.t. } \forall\, 1 \le i \le n, \quad \ell_X(h(x_i), y) \le \ell_{X^*}(\phi(x_i^*)).$$

The following optimization problem is a simplified and a generalized version of (2):

$$\min_{h \in \mathcal{H}, \phi \in \Phi} \quad \sum_{i=1}^{n}\ell_{X^*}(\phi(x_i^*), y_i) \tag{3}$$

$$\text{s.t. } \forall\, 1 \le i \le n, \quad \ell_X(h(x_i), y_i) \le \ell_{X^*}(\phi(x_i^*), y_i), \tag{4}$$

where $\ell_X$ and $\ell_{X^*}$ are arbitrary bounded loss functions, $\mathcal{H}$ is a space of decision functions and $\Phi$ is a space of correcting functions. Let $C > 0$ be a constant (that is defined later), $[t]_+ = \max(t, 0)$ and

$$\ell'((h, \phi), (x, x^*, y)) = \frac{1}{C} \cdot \ell_{X^*}(\phi(x^*), y) + [\ell_X(h(x), y) - \ell_{X^*}(\phi(x^*), y)]_+ \tag{5}$$

be the loss of the composite hypothesis $(h, \phi)$ on the example $(x, x^*, y)$. In this paper we study the relaxation of (3):

$$\min_{h \in \mathcal{H}, \phi \in \Phi} \sum_{i=1}^{n}\ell'((h, \phi), (x_i, x_i^*, y_i)), \tag{6}$$

We refer to the learning algorithm defined by the optimization problem (6) as *empirical risk minimization with privileged information*, or abbreviated *Privileged ERM*.

The basic assumption of Privileged ERM is that if we can achieve a small loss $\ell_{X^*}(\phi(x^*), y)$ in the correcting space then we should also achieve a small loss $\ell_X(h(x), y)$ in the decision space. This assumption reflects the human learning process, where the teacher tells the learner what are the most important examples (the ones with the small loss in the correcting space) that the learner should take into account in order to find a good decision rule.

The regular empirical risk minimization (ERM) finds a hypothesis $\widehat{h} \in \mathcal{H}$ that minimizes the training error $\sum_{i=1}^{n}\ell_X(h(x_i), y_i)$. While the regular ERM directly minimizes the training error of $h$, the privileged ERM minimizes the training error of $h$ indirectly, via the minimization of the training error of the correcting function $\phi$ and the relaxation of the constraint (4).

Let $h^*$ be the best possible decision function (in terms of generalization error) in the hypothesis space $\mathcal{H}$. Suppose that for each training example $x_i$ an oracle gives us the value of the loss $\ell_X(h^*(x_i), y_i)$. We use these fixed losses instead of $\ell_{X^*}(\phi(x_i^*), y_i)$ and find $h$ that satisfies the following system of inequalities:

$$\forall\, 1 \le i \le n, \ \ell_X(h(x_i), y_i) \le \ell_X(h^*(x_i), y_i). \tag{7}$$

We denote the learning algorithm defined by (7) as *OracleERM*. A straightforward generalization of the proof of Proposition 1 of [9] shows that the generalization error of the hypothesis $\widehat{h}$ found by OracleERM converges to the one of $h^*$ with the rate of $1/n$. This rate is much faster than the worst-case convergence rate $1/\sqrt{n}$ of the regular ERM [3].

In this paper we consider more realistic setting, when the above oracle is not available. Our subsequent derivations rely heavily on the following definition:

**Definition 1.1** *A decision function $h$ is uniformly better than the correcting function $\phi$ if for any example $(x, x^*, y)$ that has non-zero probability, $\ell_{X^*}(\phi(x_i^*), y_i) \geq \ell_X(h(x_i), y_i)$.*

Given a space $\mathcal{H}$ of decision functions and a space $\Phi$ of correcting functions we define

$$\overline{\Phi} = \{\phi \in \Phi \mid \exists h \in \mathcal{H} \text{ that is uniformly better than } \phi\}.$$

Note that $\overline{\Phi} \subseteq \Phi$ and $\overline{\Phi}$ does not contain correcting functions that are too good for $\mathcal{H}$. Our results are based on the following two assumptions:

**Assumption 1.2** $\overline{\Phi} \neq \emptyset$.

This assumption is not restrictive, since it only means that the optimization problem (3) of Privileged ERM has a feasible solution when the training size goes to infinity.

**Assumption 1.3** *There exists a correcting function $\overline{\phi} \in \overline{\Phi}$, such that for any $(x, x^*, y)$ that has non-zero probability, $\ell_X(h^*(x_i), y_i) = \ell_{X^*}(\overline{\phi}(x_i^*), y_i)$.*

Put it another way, we assume the existence of correcting function in $\Phi$ that mimics the losses of $h^*$.

Let $r$ be a learning rate of the Privileged ERM when it is ran over the joint $X \times X^*$ space with the space of decision and correcting functions $\mathcal{H} \times \Phi$. We develop an upper bound for the risk of the decision function found by Privileged ERM. Under the above assumptions this bound converges to $h^*$ with the same rate $r$. This implies that if the correcting space is good, so that the Privileged ERM in the joint $X \times X^*$ space has a fast learning rate (e.g $1/n$), then the Privileged ERM will have the same fast learning rate (e.g. the same $1/n$) in the decision space. That is true even if the decision space is hard and the regular ERM in the decision space has a slow learning rate (e.g. $1/\sqrt{n}$). We illustrate this result with the artificial learning problem, where the regular ERM in the decision space can not learn with the rate faster than $1/\sqrt{n}$, but the correcting space is good and Privileged ERM learns in the decision space with the rate of $1/n$.

The paper has the following structure. In Section 2 we give additional definitions. In Section 3 we review the existing risk bounds that are used to derive our results. Section 4 contains the proof of the risk bound for Privileged ERM. In Section 5 we show an example when Privileged ERM is provably better than the regular ERM. We conclude and give the directions for future research in Section 6. Due to the space constraints, most of the proofs appear in the supplementary material.

**Previous work**
The first attempt of theoretical analysis of LUPI was done by Vapnik and Vashist [9]. In addition to the analysis of learning with oracle (mentioned above), they considered the algorithm, which is close, but different from Privileged ERM. They developed a risk bound (Proposition 2 in [9]) for the decision function found by their algorithm. This bound also applies to Privileged ERM. The bound of [9] is tailored to the classification setting, with $0/1$-loss functions in the decision and the correcting space. By contrast, our bound holds for any bounded loss functions and allows the loss functions $\ell_X$ and $\ell_{X^*}$ to be different. The bound of [9] depends on generalization error of the correcting function $\widehat{\phi}$ found by Privileged ERM. Vapnik and Vashist [9] concluded that if we could bound the convergence rate of $\widehat{\phi}$ then this bound will imply the bound on the convergence rate of the decision function found by their algorithm.

## 2   Definitions

The triple $(x, x^*, y)$ is sampled from the distribution $\mathcal{D}$, which is unknown to the learner. We denote by $\mathcal{D}_X$ the marginal distribution over $(x, y)$ and by $D_{X^*}$ the marginal distribution over $(x^*, y)$. The distribution $\mathcal{D}_X$ is given by the nature and the distribution $\mathcal{D}_{X^*}$ is constructed by the teacher. The spaces $\mathcal{H}$ and $\Phi$ of decision and correcting functions are chosen by learner.

Let $R(h) = \mathrm{E}_{(x,y)\sim \mathcal{D}_X}\{\ell_X(h(x),y)\}$ and $R(\phi) = \mathrm{E}_{(x^*,y)\sim \mathcal{D}_{X^*}}\{\ell_{X^*}(\phi(x^*),y)\}$ be the generalization errors of the decision function $h$ and the correcting function $\phi$ respectively. We assume that the loss functions $\ell_X$ and $\ell_{X^*}$ have range $[0,1]$. This assumption can be satisfied by any bounded loss function by simply dividing it by its maximal value. We denote by $h^* = \arg\min_{h\in\mathcal{H}} R(h)$ and $\phi^* = \arg\min_{\phi\in\Phi} R(\phi)$ the decision and the correction function with the minimal generalization error w.r.t. the loss functions $\ell_X$ and $\ell_{X^*}$. Also, we denote by $\ell_{01}$ the $0/1$ loss, by $R_{01}(h) = \mathrm{E}_{(x,y)\sim \mathcal{D}_X}\{\ell_{01}(h(x),y)\}$ the generalization error of $h$ w.r.t. the $0/1$ loss and by $h_{01}^* = \arg\min_{h\in\mathcal{H}} R_{01}(h)$ the decision function in $\mathcal{H}$ with the minimal generalization $0/1$ error. Let $R_n'(h,\phi) = \frac{1}{n}\sum_{i=1}^n \ell'((h,\phi),(x_i,x_i^*,y_i))$ and

$$R'(h,\phi) = \mathrm{E}_{(x,x^*,y)\sim\mathcal{D}}\{\ell'((h,\phi),(x,x^*,y))\} \tag{8}$$

be respectively empirical and generalization errors of the hypothesis $(h,\phi)$ w.r.t. the loss function $\ell'$. We denote by $(\widehat{h},\widehat{\phi}) = \arg\min_{(h,\phi)\in\mathcal{H}\times\Phi} R_n'(h,\phi)$ the empirical risk minimizer and by

$$(h',\phi') = \arg\min_{(h,\phi)\in\mathcal{H}\times\Phi} R'(h,\phi)$$

the minimizer of the generalization error w.r.t. the loss function $\ell'$. Note that in general $h^*$ can be different from $h'$, and also $\phi'$ can be different from $\phi^*$.

Let

$$(\mathcal{H},\Phi) = \{(h,\phi)\in\mathcal{H}\times\Phi \mid h \text{ is uniformly better than } \phi\}.$$

By Assumption 1.2, $(\mathcal{H},\Phi)\neq\emptyset$. We will use additional technical assumption:

**Assumption 2.1** *There exists a constant $A > 0$ such that*

$$\inf\left\{\mathrm{E}_{(x,x^*,y)\sim\mathcal{D}}\left\{[\ell_X(h(x),y) - \ell_{X^*}(\phi(x^*),y)]_+\right\} \mid (h,\phi)\notin(\mathcal{H},\Phi),\ R(\phi) < R(\overline{\phi})\right\} \geq A.$$

This assumption is satisfied, for example, in the classification setting when $\ell_X$ and $\ell_{X^*}$ are $0/1$ loss functions and the probability density function $p(x,x^*,y)$ of the underlying distribution $\mathcal{D}$ is bounded away from zero for all points with nonzero probability. In this case $A \geq \inf\{p(x,x^*,y) \mid (x,x^*,y) \text{ such that } p(x,x^*,y)\neq 0\}$.

The following lemma (proved in Appendix A in the full version of the paper) shows that for sufficiently large $C$ the optimization problems (3) and (6) are asymptotically (when $n\to\infty$) equivalent:

**Lemma 2.2** *Suppose that Assumptions 1.2, 1.3 and 2.1 hold true. Then there exists a finite $C_1\in\mathbb{R}$ such that for any $C\geq C_1$, $(h',\phi')\in(\mathcal{H},\Phi)$. Moreover, $h' = h^*$ and $\phi' = \overline{\phi}$.*

In all our subsequent derivations we assume that $C$ has a finite value for which (3) and (6) are equivalent. Later on we will show how we choose the value of $C$ that optimizes the forthcoming risk bound.

The risk bounds presented in this paper are based on VC-dimension of various function classes. While the definition of VC-dimension for binary functions is well-known in the learning community, the one for the real-valued functions is less known and we review it here. Let $\mathcal{F}$ be a set of real-valued functions $f : \mathcal{S}\to\mathbb{R}$ and $\overline{\mathcal{T}}(\mathcal{F}) = \{(x,t)\in\mathcal{S}\times\mathbb{R} \mid \exists f\in\mathcal{F} \text{ s.t. } 0\leq|f(x)|\leq t\}$. We say that the set $\mathcal{T} = \{(x_i,t_i)\}_{i=1}^{|\mathcal{T}|}\subseteq\overline{\mathcal{T}}(\mathcal{F})$ is *shattered* by $\mathcal{F}$ if for any $\mathcal{T}'\subseteq\mathcal{T}$ there exists a function $f\in\mathcal{F}$ such that for any $(x_i,t_i)\in\mathcal{T}'$, $|f(x_i)|\leq t_i$ and for any $(x_i,t_i)\in\mathcal{T}\setminus\mathcal{T}'$, $|f(x_i)| > t_i$. The VC-dimension of $\mathcal{F}$ is defined as a VC-dimension of the set $\overline{\mathcal{T}}(\mathcal{F})$, namely the maximal size of the set $\mathcal{T}\subseteq\overline{\mathcal{T}}(\mathcal{F})$ that is shattered by $\mathcal{F}$.

## 3 Review of existing excess risk bounds with fast convergence rates

We derive our risk bounds from generic excess risk bounds developed by Massart and Nedelec [6] and generalized by Gine and Koltchinskii [4] and Koltchinkii [5]. In this paper we use the version of the bounds given in [4] and [5].

Let $\mathcal{F}$ be a space of hypotheses $f : \mathcal{S}\to\mathcal{S}'$, $\ell : \mathcal{S}'\times\{-1,+1\}\to\mathbb{R}$ be a real-valued loss function such that $0\leq\ell(f(x),y)\leq 1$ for any $f\in\mathcal{F}$ and any $(x,y)$. Let $f^* =$

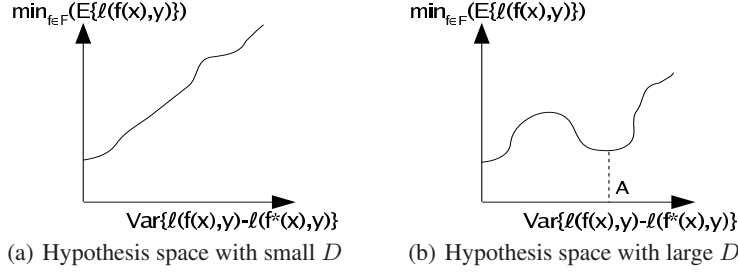

(a) Hypothesis space with small $D$      (b) Hypothesis space with large $D$

Figure 1: Visualization of the hypothesis spaces. The horisontal axis measures the distance (in terms of the variance) between hypothesis $f$ and the best hypothesis $f^*$ in $\mathcal{F}$. The vertical axis is the minimal error of hypotheses in $\mathcal{F}$ with the fixed distance from $f^*$. Note that the error function displayed in graphs can be non-continuous. The large value of $D$ in the hypothesis space in graph (b) is caused by hypothesis $A$, which is significantly different from $f^*$ but has nearly-optimal error.

$\arg\min_{f\in\mathcal{F}} \mathrm{E}_{(x,y)}\{\ell(f(x),y)\}$, $\widehat{f}_n = \arg\min_{f\in\mathcal{F}} \sum_{i=1}^{n} \ell(f(x_i),y_i)$ and $D > 0$ be a constant such that for any $f \in \mathcal{F}$,

$$\mathrm{Var}_{(x,y)}\{\ell(f(x),y) - \ell(f^*(x),y)\} \leq D \cdot \mathrm{E}_{(x,y)}\{\ell(f(x),y) - \ell(f^*(x),y)\}. \qquad (9)$$

This condition is a generalization of Tsybakov's low-noise condition [7] to arbitrary loss functions and arbitrary hypothesis spaces.

The constant $D$ in (9) characterizes the error surface of the hypothesis space $\mathcal{F}$. Suppose that $\mathrm{E}_{(x,y)}\{\ell(f(x),y) - \ell(f^*(x),y)\}$ is very small, namely $f$ is nearly optimal. If $f$ is almost the same as $f^*$ then the variance in the left hand side of (9), as well as the value of $D$, will be small. But if $f$ differs significantly from $f^*$ then the variance in the left hand side of (9), as well as the value of $D$, will be large. Thus, if we take the variance in the left hand side of (9) as a measure of distance between $f$ and $f^*$ then the hypothesis spaces with large and small $D$ can be visualized as shown in Figure 1.

Let $V$ be a VC-dimension of $\mathcal{F}$. The following theorem is a straightforward generalization of Theorem 5.8 in [5].

**Theorem 3.1 ([5])** *There exists a constant $K > 0$ such that if $n > V \cdot D^2$ then for any $\delta > 0$, with probability of at least $1 - \delta$*

$$\mathrm{E}_{(x,y)}\{\ell(\widehat{f}(x),y)\} \leq \mathrm{E}_{(x,y)}\{\ell(f^*(x),y)\} + \frac{KD}{n}\left(V\log\frac{n}{VD^2} + \ln\frac{1}{\delta}\right). \qquad (10)$$

Let $B = (V\log n + \log(1/\delta))/n$. If the condition of Theorem 3.1 does not hold, namely if $n \leq V \cdot D^2$ then we can use the following fallback risk bound:

**Theorem 3.2 ([1, 8])** *There exists a constant $K'$ such that for any $\delta > 0$, with probability of at least $1 - \delta$,*

$$\mathrm{E}_{(x,y)}\{\ell(\widehat{f}(x),y)\} \leq \mathrm{E}_{(x,y)}\{\ell(f^*(x),y)\} + K'\left(\sqrt{\mathrm{E}_{(x,y)}\{\ell(f^*(x),y)\}B} + B\right). \qquad (11)$$

**Definition 3.3** *Let $T = T(\mathrm{E}_{(x,y)}\{\ell(f^*(x),y)\}, V, \delta)$ be a constant such that for all $n < T$ it holds that $\mathrm{E}_{(x,y)}\{\ell(f^*(x),y)\} < B$.*

For $n \leq T$ the bound (11) has a convergence rate of $1/n$, and for $n > T$ the bound (11) has a convergence rate of $1/\sqrt{n}$. The main difference between (10) and (11) is the fast convergence rate of $1/n$ vs. the slow one of $1/\sqrt{n}$ in the regime of $n > \max(T, V \cdot D^2)$. By Theorem 3.1, starting from $n > n(D) = V \cdot D^2$ we always have the convergence rate of $1/n$. Thus, the smaller value of $D$, the smaller will be the threshold $n(D)$ for obtaining the fast convergence rate of $1/n$.

## 4 Upper Risk Bound

For any $C \geq 1$, any $(x, x^*, y)$, any $h \in \mathcal{H}$ and $\phi \in \Phi$, and any loss functions $\ell_X$ and $\ell_{X^*}$,

$$\ell_X(h(x),y) \leq \ell_{X^*}(\phi(x^*),y) + C\left[\ell_X(h(x),y) - \ell_{X^*}(\phi(x^*),y)\right]_+.$$

Hence, using (5) we obtain that

$$R(\widehat{h}) = \mathrm{E}_{(x,y)}\{\ell_X(\widehat{h}(x), y)\} \leq C \cdot \mathrm{E}_{(x^*,y)}\left\{\ell'((\widehat{h}, \widehat{\phi}), (x, x^*, y))\right\} = C \cdot R'(\widehat{h}, \widehat{\phi}). \qquad (12)$$

Let $\ell_1(h, h^*, x, y) = \ell_X(h(x), y) - \ell_X(h^*(x), y)$ and $D_{\mathcal{H}} \geq 0$ be a constant such that for any $h \in \mathcal{H}$

$$D_{\mathcal{H}} \cdot \mathrm{E}_{(x,y)}\{\ell_1(h, h^*, x, y)\} \geq \mathrm{Var}_{(x,y)}\{\ell_1(h, h^*, x, y)\}. \qquad (13)$$

Similarly, let $\ell_2(h, h', \phi, \phi', x, x^*, y) = \ell'((h, \phi), (x, x^*, y)) - \ell'((h', \phi'), (x, x^*, y))$ and $D_{\mathcal{H},\Phi} \geq 0$ be a constant such that for all $(h, \phi) \in \mathcal{H} \times \Phi$,

$$D_{\mathcal{H},\Phi} \cdot \mathrm{E}_{(x,x^*,y)}\{\ell_2(h, h', \phi, \phi', x, x^*, y)\} \geq \mathrm{Var}_{(x,x^*,y)}\{\ell_2(h, h', \phi, \phi', x, x^*, y)\}. \qquad (14)$$

Let $\mathcal{L}(\mathcal{H}, \Phi) = \{\ell'((h, \phi), (\cdot, \cdot, \cdot)) \mid h \in \mathcal{H}, \; \phi \in \Phi\}$ be a set of the loss functions $\ell'$ corresponding to hypotheses from $\mathcal{H} \times \Phi$ and $V_{\mathcal{L}(\mathcal{H},\Phi)}$ be a VC-dimension of $\mathcal{L}(\mathcal{H}, \Phi)$. Similarly, let $\mathcal{L}(\mathcal{H}) = \{\ell_X(h(\cdot), \cdot) \mid h \in \mathcal{H}\}$ and $\mathcal{L}(\Phi) = \{\ell_{X^*}(\phi(\cdot), \cdot) \mid \phi \in \Phi\}$ be the sets of loss functions that correspond to the hypotheses in $\mathcal{H}$ and $\Phi$, and $V_{\mathcal{L}(\mathcal{H})}$ and $V_{\mathcal{L}(\Phi)}$ be VC dimensions of $\mathcal{L}(\mathcal{H})$ and $\mathcal{L}(\Phi)$ respectively. Note that if $\ell_X = \ell_{01}$ then $V_{\mathcal{L}(\mathcal{H})}$ is also a VC-dimension of $\mathcal{H}$ (the same holds also for $V_{\mathcal{L}(\Phi)}$).

**Lemma 4.1** $V_{\mathcal{L}(\mathcal{H},\Phi)} = V_{\mathcal{L}(\mathcal{H})} + V_{\mathcal{L}(\Phi)}$.

**Proof** See Appendix C in the full version of the paper.

We apply Theorem 3.1 to the hypothesis space $\mathcal{H} \times \Phi$ and the loss function $\ell'((h, \phi), (x, x^*, y))$ and obtain that there exists a constant $K > 0$ such that if $n > V_{\mathcal{L}(\mathcal{H},\Phi)} \cdot D_{\mathcal{H},\Phi}^2$ then for any $\delta > 0$, with probability at least $1 - \delta$

$$R'(\widehat{h}, \widehat{\phi}) \leq R'(h', \phi') + \frac{KD_{\mathcal{H},\Phi}}{n}\left(V_{\mathcal{L}(\mathcal{H},\Phi)} \ln \frac{n}{V_{\mathcal{L}(\mathcal{H},\Phi)}D_{\mathcal{H},\Phi}^2} + \ln \frac{1}{\delta}\right).$$

Using (12) we obtain that

$$R(\widehat{h}) \leq C \cdot R'(h', \phi') + \frac{CKD_{\mathcal{H},\Phi}}{n}\left(V_{\mathcal{L}(\mathcal{H},\Phi)} \ln \frac{n}{V_{\mathcal{L}(\mathcal{H},\Phi)}D_{\mathcal{H},\Phi}^2} + \ln \frac{1}{\delta}\right). \qquad (15)$$

It follows from Assumption 1.3 and Lemma 2.2 that

$$R'(h', \phi') = \frac{1}{C}R(\phi') = \frac{1}{C}R(\overline{\phi}) = \frac{1}{C}R(h^*). \qquad (16)$$

We substitute (16) into (15) and obtain that there exists a constant $K > 0$ such that if $n > V_{\mathcal{L}(\mathcal{H},\Phi)} \cdot D_{\mathcal{H},\Phi}^2$ then for any $\delta > 0$, with probability at least $1 - \delta$,

$$R(\widehat{h}) \leq R(h^*) + \frac{CKD_{\mathcal{H},\Phi}}{n}\left(V_{\mathcal{L}(\mathcal{H},\Phi)} \ln \frac{n}{V_{\mathcal{L}(\mathcal{H},\Phi)}D_{\mathcal{H},\Phi}^2} + \ln \frac{1}{\delta}\right).$$

We bound $V_{\mathcal{H},\Phi}$ by Lemma 4.1 and obtain our final risk bound, that is summarized in the following theorem:

**Theorem 4.2** *Suppose that Assumptions 1.2, 1.3 and 2.1 hold. Let $D_{\mathcal{H},\Phi}$ be as defined in (14), $C_1$ be as defined in Lemma 2.2, and $\overline{V}_{\mathcal{L}(\mathcal{H},\Phi)} = V_{\mathcal{L}(\mathcal{H})} + V_{\mathcal{L}(\Phi)}$. Suppose that $C > C_1$ and $n > \overline{V}_{\mathcal{L}(\mathcal{H},\Phi)} \cdot D_{\mathcal{H},\Phi}^2$. Then for any $\delta > 0$ with probability of at least $1 - \delta$,*

$$R(\widehat{h}) \leq R(h^*) + \frac{CKD_{\mathcal{H},\Phi}}{n}\left(\overline{V}_{\mathcal{L}(\mathcal{H},\Phi)} \ln \frac{n}{\overline{V}_{\mathcal{L}(\mathcal{H},\Phi)} \cdot D_{\mathcal{H},\Phi}^2} + \ln \frac{1}{\delta}\right), \qquad (17)$$

*where $K > 0$ is a constant.*

According to this bound, $R(\widehat{h})$ converges to $R(h^*)$ with the rate of $1/n$. If Assumption 1.3 does not hold then it is easy to see that we obtain the same bound as (17), but with $R(h^*)$ replaced by $R(\phi')$. In this case the upper bound on $R(\widehat{h})$ converges to $R(\phi')$ with the rate of $1/n$.

We now provide further analysis of the risk bound (17). Let $\ell_3(\phi, \phi', x^*, y) = \ell_{X^*}(\phi(x^*), y) - \ell_{X^*}(\phi'(x^*), y)$ and $D_\Phi \geq 0$ be a constant such that for any $\phi \in \overline{\Phi}$,

$$D_\Phi \cdot \mathrm{E}_{(x^*,y)}\left\{\ell_3(\phi, \phi', x^*, y)\right\} \geq \mathrm{Var}_{(x^*,y)}\left\{\ell_3(\phi, \phi', x^*, y)\right\}. \tag{18}$$

Similarly, let $D'_{\mathcal{H},\Phi} \geq 0$ be a constant such that for all $(h, \phi) \in (\mathcal{H} \times \Phi) \setminus (\mathcal{H}, \Phi)$,

$$D'_{\mathcal{H},\Phi}\mathrm{E}_{(x,x^*,y)}\left\{\ell_2(h, h', \phi, \phi', x, x^*, y)\right\} \geq \mathrm{Var}_{(x,x^*,y)}\left\{\ell_2(h, h', \phi, \phi', x, x^*, y)\right\}.$$

**Lemma 4.3** $D_{\mathcal{H},\Phi} \leq \max\left(D_\Phi/C, D'_{\mathcal{H},\Phi}\right)$.

**Proof** See Appendix B in the full version of the paper.

By Lemma 4.3, $C \cdot D_{\mathcal{H},\Phi} \leq \max(D_\Phi, C \cdot D'_{\mathcal{H},\Phi})$. Since the loss function $\ell_2$ depends on $C$, the constant $D'_{\mathcal{H},\Phi}$ depends on $C$ too. Thus, ingoring the left-hand logarithmic term in (17), the optimal value of $C$ is the one that is larger that $C_1$ and minimizes $C \cdot D'_{\mathcal{H},\Phi}$. We now show that such minimum indeed exists. By the definition of the loss function $\ell_2$,

$$0 < \lim_{C \to \infty} \sup_{(h,\phi) \in (\mathcal{H} \times \Phi) \setminus (\mathcal{H}, \Phi)} \left\{ \frac{\mathrm{Var}_{(x,x^*,y)}\left\{\ell_2(h, h', \phi, \phi', x, x^*, y)\right\}}{\mathrm{E}_{(x,x^*,y)}\left\{\ell_2(h, h', \phi, \phi', x, x^*, y)\right\}} \right\} \leq 1. \tag{19}$$

Therefore for very large $C$ it holds that $0 < s \leq D'_{\mathcal{H},\Phi} \leq 1$, where $s$ is the value of the above limit. Consequently $\lim_{C \to \infty} C \cdot D'_{\mathcal{H},\Phi} = \infty$. Since the function $g(C) = C \cdot D'_{\mathcal{H},\Phi}$ is continuous and finite in $C = C_1$, there exists a point $C = C^* \in [C_1, \infty)$ that minimizes it.

# 5    When Privileged ERM is provably better than the regular ERM

We show an example that demonstrates the difference between the emprical risk minimization in $X$ space and empirical risk minimization with privileged information in the joint $X \times X^*$ space. In particular, we show in this example that for not too small training sizes (as specified by the conditions of Theorems 11 and 4.2) the learning rate of the regular ERM in $X$ space is $1/\sqrt{n}$ while the learning rate of the privileged ERM in the joint $X \times X^*$ space is $1/n$.

We consider the classification setting and all loss functions in our example are $0/1$ loss. Let $\overline{\mathcal{D}}_X = \{\mathcal{D}_X(\epsilon)|0 < \epsilon < 0.1\}$ be an infinite family of distributions of examples in $X$ space. All distributions in $\overline{\mathcal{D}}_X$ have non-zero support in four points, denoted by $X_1$, $X_2$, $X_3$ and $X_4$. We assume that these points lie on a 1-dimensional line, as shown in Figure 2(a). Figure 2(a) also shows the probability mass of each point in the distribution $\mathcal{D}_X(\epsilon)$. The hypothesis space $\mathcal{H}$ consists of hypotheses $h_t(x) = \mathrm{sign}(x - t)$ and $h'_t = -\mathrm{sign}(x - t)$. The best hypothesis in $\mathcal{H}$ is $h'_1$ and its generalization error is $1/4 - 2\epsilon$. The hypothesis space $\mathcal{H}$ contains also a hypothesis $h'_3$, which is slightly worse than $h'_1$ and has generalization error of $1/4 + \epsilon$. It can be verified that for a fixed $\mathcal{D}_X(\epsilon)$ and $\mathcal{H}$ the constant $D_\mathcal{H}$ (defined in equation (13)) is

$$D_\mathcal{H} = 1/(6\epsilon) - (1/3) - \epsilon \leq 1/(6\epsilon). \tag{20}$$

Note that the inequality in (20) is very tight since $\epsilon$ can be arbitrary small. The VC-dimension $V_\mathcal{H}$ of $\mathcal{H}$ is 2. Suppose that $\epsilon$ is sufficiently small such that $V_\mathcal{H} \cdot D_\mathcal{H}^2 > T(1/4 - 2\epsilon, V_\mathcal{H}, \delta)$, where the function $T(\cdot, \cdot, \cdot)$ is defined in Definition 3.3. In order to use the risk bound (10) with our $\mathcal{D}_X$ and $\mathcal{H}$, the condition

$$n > V_\mathcal{H} \cdot D_\mathcal{H}^2 = 1/(18\epsilon^2) \tag{21}$$

should be satisfied. But since $\epsilon$ can be very small, the condition (21) is not satisfied for a large range of $n$'s. Hence, according to (11), for distributions $\mathcal{D}_X(\epsilon)$ that satisfy $T(1/4 - 2\epsilon, 2, \delta) \leq \frac{1}{18\epsilon^2}$ we obtain that $R_{01}(\widehat{h})$ converges to $R_{01}(h^*)$ with the rate of at least $1/\sqrt{n}$.

The following lower bound shows that $R_{01}(\widehat{h})$ converges to $R_{01}(h^*)$ with the rate of at most $1/\sqrt{n}$.

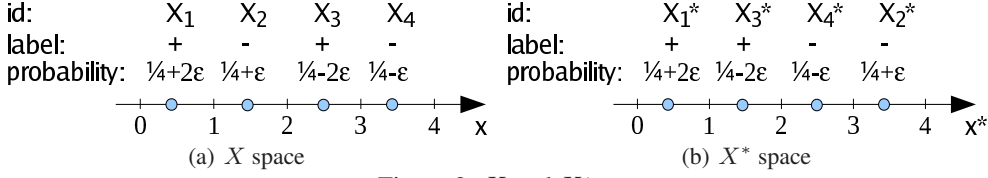

(a) $X$ space            (b) $X^*$ space

Figure 2: $X$ and $X^*$ spaces.

**Lemma 5.1** *Suppose that $\epsilon < 1/16$. Let $\delta_n = \exp(-20n\epsilon^2)$. Then for any $n > 256$, with probability at least $\delta_n$,*

$$R_{01}(\widehat{h}) - R_{01}(h^*) \geq \sqrt{\ln(1/\delta_n)/(20n)}.$$

By combining upper and lower bounds we obtain that the convergence rate of $R_{01}(\widehat{h})$ to $R_{01}(h^*)$ is exactly $1/\sqrt{n}$. The proof of the lower bound appears in Appendix D in the full version of the paper.

Suppose that the teacher constructed the distribution $\mathcal{D}_{X^*}(\epsilon)$ of examples in $X^*$ space in the following way. $\mathcal{D}_{X^*}(\epsilon)$ has non-zero support in four points, denoted by $X_1^*$, $X_2^*$, $X_3^*$ and $X_4^*$, that lie on a 1-dimensional line, as shown in Figure 2(b). Figure 2(b) shows the probability mass of each point in $X^*$ space. We assume that the joint distribution $(X, X^*)$ has non-zero support only on points $(X_1, X_1^*)$, $(X_2, X_2^*)$, $(X_3, X_3^*)$ and $(X_4, X_4^*)$. The hypothesis space $\Phi$ consists of hypotheses $\phi_t(x) = \text{sign}(x^* - t)$ and $\phi_t' = -\text{sign}(x^* - t)$. The best hypothesis in $\Phi$ is $\phi_2'$ and its generalization error is 0. However there is no $h \in \mathcal{H}$ that is uniformly better than $\phi_2'$. The best hypothesis in $\Phi$, among those that have uniformly better hypothesis in $\mathcal{H}$, is $\phi_1'$ and its generalization error is $1/4 - 2\epsilon$. $h_1'$ is uniformly better than $\phi_1'$. It can be verified that for such $\mathcal{D}_{X^*}(\epsilon)$ and $\Phi$ the constant $D_\Phi$ (defined in equation (18)) is

$$D_\Phi = (11/16 - 3\epsilon - 4\epsilon^2)/(1/4 + 2\epsilon) \leq 2.75. \tag{22}$$

Note that the inequality in (22) is very tight since $\epsilon$ can be arbitrary small. Moreover, it can be verified that $C$ that minimizes $C \cdot D_{\mathcal{H},\Phi}'$ is $C^* = 2.6$. For $C = C^*$ it holds that $D_{\mathcal{H},\Phi}' = 1.71$ and $D_\Phi/C = 1.06$. It is easy to see that our example satisfies Assumptions 1.2 and 1.3 (the last assumption is satisfied with $\overline{\phi} = -\phi_1'$). Also, it can be verified that Assumption 2.1 is satisfied with $A = 1/4 - 2\epsilon$ and $C_1 = 1.1 < C^*$ satisfies Lemma 2.2. The VC-dimension of $\Phi$ is 2. Hence by Theorem 4.2 and Lemma 4.3, if $n > (2 + 2) \cdot 1.71^2 = 11.7$ then $R_{01}(\widehat{h})$ converges to $R_{01}(h^*)$ with the rate of at least $1/n$. Since our bounds on $D_\Phi$ and $D_{\mathcal{H},\Phi}'$ are independent of $\epsilon$, the convergence rate of $1/n$ holds for any distribution in $\overline{\mathcal{D}}_X$.

We obtained that for $11.7 < n \leq \frac{1}{18\epsilon^2}$ the upper bound (17) converges to $R_{01}(h^*)$ with the rate of $1/n$, while the upper bound (11) converges to $R_{01}(h^*)$ with the rate of $1/\sqrt{n}$. This improvement was possible due to teacher's construction of $\mathcal{D}_{X^*}(\epsilon)$ and learner's choice of $\Phi$. The hypothesis $h_3'$ caused the value of $D_\mathcal{H}$ to be large and thus prevented us from $1/n$ convergence rate for a large range of $n$'s. We constructed $\mathcal{D}_{X^*}(\epsilon)$ and $\Phi$ in such a way that $\Phi$ does not have a hypothesis $\phi$ that has exactly the same dichotomy as the bad hypothesis $h_3'$. With such construction any $\phi \in \Phi$, such that $h_3'$ is uniformly better than $\phi$, has generalization error significantly larger than the one of $h_3'$. For example, the best hypothesis in $\Phi$ for which $h_3'$ is uniformly better, is $\phi_0$ and its generalization error is $1/2$.

## 6    Conclusions

We formulated the algorithm of empirical risk minimization with privileged information and derived the risk bound for it. Our risk bound outlines the conditions for the correcting space that, if satisfied, will allow fast learning in the decision space, even if the original learning problem in the decision space is very hard. We showed an example where the privileged information provably significantly improves the learning rate.

In this paper we showed that the good correcting space can improve the learning rate from $1/\sqrt{n}$ to $1/n$. But, having the good correcting space, can we achieve a learning rate faster than $1/n$? Another intersting problem is to analyze Privileged ERM when the learner does not completely trust the teacher. This condition translates to the constraint $\ell_X(h(x), y) \leq \ell_{X^*}(\phi(x^*), y) + \epsilon$ in (3) and the term $[\ell_X(h(x), y) - \ell_{X^*}(\phi(x^*), y)]_+$ in (6), where $\epsilon \geq 0$ is a hyperparameter. Finally, the important direction is to develop risk bounds for SVM+ (which is a regularized version of Privileged ERM) and show when it is provably better than SVM.

# References

[1] S. Boucheron, O. Bousquet, and G. Lugosi. Theory of classification: a survey of some recent advances. *ESAIM: Probability and Statistics*, 9:329–375, 2005.

[2] C. Cortes and V. Vapnik. Support-vector networks. *Machine Learning*, 20(3):273–297, 1995.

[3] L. Devroye and G. Lugosi. Lower bounds in pattern recognition and learning. *Pattern Recognition*, 28(7):1011–1018, 1995.

[4] E. Gine and V. Koltchinskii. Concentration inequalities and asymptotic resutls for ratio type empirical processes. *Annals of Probability*, 34(3):1143–1216, 2006.

[5] V. Koltchinskii. 2008 Saint Flour lectures: Oracle inequalities in empirical risk minimization and sparse recovery problems, 2008. Available at fodava.gatech.edu/files/reports/FODAVA-09-17.pdf.

[6] P. Massart and E. Nedelec. Risk bounds for statistical learning. *Annals of Statistics*, 34(5):2326–2366, 2006.

[7] A. Tsybakov. Optimal aggregation of classifiers in statistical learning. *Annals of Statistics*, 32(1):135–166, 2004.

[8] V. Vapnik. *Estimation of dependencies based on empirical data*. Springer–Verlag, 2nd edition, 2006.

[9] V. Vapnik and A. Vashist. A new learning paradigm: Learning using privileged information. *Neural Networks*, 22(5-6):544–557, 2009.

[10] V. Vapnik, A. Vashist, and N. Pavlovich. Learning using hidden information: Master class learning. In *Proceedings of NATO workshop on Mining Massive Data Sets for Security*, pages 3–14. 2008.

